# Linking motor learning to function approximation: Learning in an unlearnable force field

**Opher Donchin and Reza Shadmehr**

Dept. of Biomedical Engineering

Johns Hopkins University, Baltimore, MD 21205

Email: opher@bme.jhu.edu, reza@bme.jhu.edu

## Abstract

Reaching movements require the brain to generate motor commands that rely on an internal model of the task's dynamics. Here we consider the errors that subjects make early in their reaching trajectories to various targets as they learn an internal model. Using a framework from function approximation, we argue that the sequence of errors should reflect the process of gradient descent. If so, then the sequence of errors should obey hidden state transitions of a simple dynamical system. Fitting the system to human data, we find a surprisingly good fit accounting for 98% of the variance. This allows us to draw tentative conclusions about the basis elements used by the brain in transforming sensory space to motor commands. To test the robustness of the results, we estimate the shape of the basis elements under two conditions: in a traditional learning paradigm with a consistent force field, and in a random sequence of force fields where learning is not possible. Remarkably, we find that the basis remains invariant.

## 1   Introduction

It appears that in constructing the motor commands to guide the arm toward a target, the brain relies on an internal model (IM) of the dynamics of the task that it learns through practice [1]. The IM is presumably a system that transforms a desired limb trajectory in sensory coordinates to motor commands. The motor commands in turn create the complex activation of muscles necessary to cause action. A major issue in motor control is to infer characteristics of the IM from the actions of subjects.

Recently, we took a first step toward mathematically characterizing the IM's representation in the brain [2]. We analyzed the sequence of errors made by subjects on successive movements as they reached to targets while holding a robotic arm. The robot produced a force field and subjects learned to compensate for the field (presumably by constructing an IM) and eventually produced straight movements within the field. Our analysis sought to draw conclusions about the structure of the IM from the sequence of errors generated by the subjects. For instance, in a

velocity-dependent force field (such as the fields we use), the IM must be able to encode velocity in order to anticipate the upcoming force. We hoped that the effect of errors in one direction on subsequent movements in other directions would give information about the width of the elements which the IM used in encoding velocity. For example, if the basis elements were narrow, then movements in a given direction would result in little or no change in performance in neighboring directions. Wide basis elements would mean appropriately larger effects.

We hypothesized that an estimate of the width of the basis elements could be calculated by fitting the time sequence of errors to a set of equations representing a dynamical system. The dynamical system assumed that error in a movement resulted from a difference between the IM's approximation and the actual environment, an assumption that has recently been corroborated [3]. The error in turn changed the IM, affecting subsequent movements:

$$
\begin{cases}
y^{(n)} = D_{k^{(n)}} F^{(n)} - z_{k^{(n)}}^{(n)} \\
z_l^{(n+1)} = z_l^{(n)} + B_{l,k^{(n)}} y^{(n)} \quad l = 1, \cdots, 8
\end{cases}
\tag{1}
$$

Here $y^{(n)}$ is the error on the $n$th movement, made in direction $k^{(n)}$ (8 possible directions); $F^{(n)}$ is the actual force experienced in the movement, and it is scaled by an arm compliance $D$ which is direction dependent; and $z_k^{(n)}$ is the current output of the IM in the direction $k$. The difference between this output and reality results in movement errors. $B$ is a matrix characterizing the effect of errors in one direction on other directions. That is, $B$ can provide the generalization function we sought. By comparing the $B$ produced by a fit to human data to the $B$s produced from simulated data (generated using a dynamical simulation of arm movements), we found that the time sequence of the subjects' errors was similar to that generated by a simulation that represented the IM with gaussian basis elements that encoded velocity with a $\sigma = 0.08$ m/sec.

But why might this dynamical system be a good model of trial-to-trial behavior in a learning paradigm? Here we demonstrate that, under reasonable assumptions, behavior in accordance with Eq. 1 can be derived within the framework of functional approximation, and that $B$ is closely related to the basis functions in the approximation process. We find that this model gives accurate fits to human data, even when the number of parameters in the model is drastically reduced. Finally, we test the prediction of Eq. 1 that learning involves simple movement-by-movement corrections to the IM, and that these variations depend only on the shape of the basis which the IM uses for representation. Remarkably, when subjects perform movements in a force field that changes randomly from one movement to the next, the pattern of errors predicts a generalization function, and therefore a set of basis elements, indistinguishable from the condition where the force field does not change. That is, "an unlearnable task is learned in exactly the same way as a learnable task."

## 2 Approach

### 2.1 The Learning Process

In the current task, subjects grip the handle of a robot and make 10cm reaching movements to targets presented visually. The robot produces a force field $\mathbf{F}(\dot{\mathbf{x}})$ proportional and perpendicular to the velocity of the hand, such as $\mathbf{F} = (0 \ 13; -13 \ 0) \cdot \dot{\mathbf{x}}$ (with $\mathbf{F}$ in Newtons and $\dot{\mathbf{x}}$ in m/s). To simulate the process of learning an IM, we assume that the IM uses scalar valued basis functions that encode velocity $\mathbf{g} = [g_1(\dot{\mathbf{x}}), \ldots, g_n(\dot{\mathbf{x}})]^T$ so that the IM's expectation of force at a desired velocity is: $\hat{\mathbf{F}}(\dot{\mathbf{x}}) = W\mathbf{g}(\dot{\mathbf{x}})$, where $W$ is a $2 \times n$ matrix [4]. To move the hand to a

target at direction $k$, a desired trajectory $\dot{\mathbf{x}}_k(t)$ is given as input to the IM, which in turn produces as output $\hat{\mathbf{F}}(\dot{\mathbf{x}}_k)$ [5, 6]. As a result, forces are experienced $\mathbf{F}(t)$ so that a force error can be calculated as $\tilde{\mathbf{F}}(t) = \mathbf{F}(t) - \hat{\mathbf{F}}(\dot{\mathbf{x}}_k(t))$. We adjust $W$ in the direction that minimizes a cost function $e$ which is simply the magnitude of the force error integrated over the entire movement:

$$ e = \frac{1}{2} \int_0^T \tilde{\mathbf{F}}(t)^T \tilde{\mathbf{F}}(t)\, dt = \frac{1}{2} \int_0^T (\mathbf{F}(t) - W\mathbf{g}(t))^T (\mathbf{F}(t) - W\mathbf{g}(t))\, dt $$

Changing $W$ to minimize this value requires that we calculate the gradient of $e$ with respect to the weights and move $W$ in the direction opposite to the gradient:

$$ (\triangledown e)_{W_{ij}} = \frac{\partial e}{\partial W_{ij}} = -\int_0^T g_j(t)\tilde{F}_i(t)\, dt $$

$$ W^{(n+1)} = W^{(n)} + \eta \int_{t=0}^T \tilde{\mathbf{F}}^{(n)}(t)\mathbf{g}(\dot{\mathbf{x}}_{k^{(n)}}(t))^T\, dt \qquad (2) $$

where $W^{(n)}$ means the $W$ matrix on the $n$th movement.

## 2.2 Deriving the Dynamical System

Our next step is to represent learning not in terms of weight changes, but in terms of changes in IM output, $\hat{\mathbf{F}}$. We do this for an arbitrary point in velocity space $\dot{\mathbf{x}}_0$ by multiplying both sides of the Eq. 2 by $\mathbf{g}(\dot{\mathbf{x}}_0)$ with the result that:

$$ \hat{\mathbf{F}}^{(n+1)}(\dot{\mathbf{x}}_0) = \hat{\mathbf{F}}^{(n)}(\dot{\mathbf{x}}_0) + \eta \int_{t=0}^T \left[\mathbf{g}(\dot{\mathbf{x}}_{k^{(n)}})^T \mathbf{g}(\dot{\mathbf{x}}_0)\right] \tilde{\mathbf{F}}^{(n)}\, dt \qquad (3) $$

Further simplification will require approximation. Because we are considering a case where the actual force, $\mathbf{F}(\dot{\mathbf{x}})$, is directly proportional to velocity, it is reasonable to make the approximation that, along a reasonably straight desired trajectory, the force error, $\tilde{\mathbf{F}}(t)$, is simply proportional to the velocity, $\tilde{\mathbf{F}}(\dot{\mathbf{x}}_{k^{(n)}}) = \tilde{\mathbf{F}} \cdot \dot{\mathbf{x}}_{k^{(n)}}$. This means that the integral of Eq. 3 is actually of the form

$$ \tilde{\mathbf{F}} \int_{t=0}^T \dot{\mathbf{x}}_{k^{(n)}}(t)\mathbf{g}(\dot{\mathbf{x}}_{k^{(n)}})^T \mathbf{g}(\dot{\mathbf{x}}_0)\, dt \qquad (4) $$

One more assumption is required to make this tractable. If we approximate the desired trajectory with a triangular function of time, and integrate only over the raising phase of the velocity curve (because the values are the same going up and going down) we can simplify the integral to an integral over speed, drawing out a constant $(2K \int_{\dot{x}=0}^{\dot{x}=\dot{x}_k(250\text{ms})} G(\dot{x}, \dot{x}_0)\, d\dot{x})$. The integral has become a function of the values of $\dot{\mathbf{x}}_{k^{(n)}}(250\text{ms})$ and $\dot{\mathbf{x}}_0$. Calling this function $B$, Eq. 4 becomes

$$ \hat{\mathbf{F}}^{(n+1)}(\dot{\mathbf{x}}_0) = \hat{\mathbf{F}}^{(n)}(\dot{\mathbf{x}}_0) + B(\dot{\mathbf{x}}_{k^{(n)}}, \dot{\mathbf{x}}_0)\tilde{\mathbf{F}}^{(n)} \qquad (5) $$

$\dot{\mathbf{x}}_0$ is arbitrary. We restrict our attention to only $\dot{\mathbf{x}}_0$ that equals the peak velocity of the desired trajectory associated with a movement direction $l$. Since we have only eight different points in velocity space to consider, $\hat{\mathbf{F}}$ can be considered an eight-valued vector, $\hat{\mathbf{F}}_l$ rather than a function $\hat{\mathbf{F}}(\dot{\mathbf{x}})$. Similarly, $B(\dot{\mathbf{x}}_l, \dot{\mathbf{x}}_k)$ will become an 8x8 matrix, $B_{l,k}$. The simpler notation allows us to write Eq. 5 as

$$ \hat{\mathbf{F}}_l^{(m+1)} = \hat{\mathbf{F}}_l^{(n)} + B_{l,k^{(n)}}\tilde{\mathbf{F}}^{(n)} \qquad l = 1, \ldots, 8 \qquad (6) $$

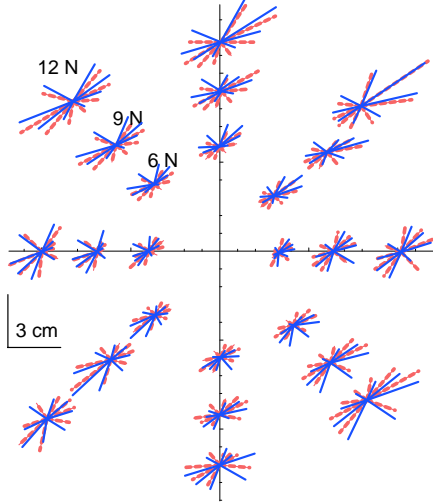

Figure 1: We performed simulations to test the approximation that displacement in arm motion at 250 msec toward a target at 10 cm is proportional to error in the force estimate made by the IM. A system of equations describing a controller, dynamics of a typical human arm, and robot dynamics [7] were simulated for a 500 msec min jerk motion to 8 targets. The simulated robot produced one of 8 force fields scaled to 3 different magnitudes, while the controller remained naïve to the field. The errors in hand motion at 250 msec were fitted to the robot forces using a single compliance matrix. Lighter dashed lines are the displacement predictions of the model, darker solid lines are the actual displacement in the simulations' movement.

One more approximation is to assume that force error $\tilde{\mathbf{F}}$ in a given movement will be proportional to position error in that movement when both are evaluated at 250ms. This approximation is justified by the data presented in Fig. 1 which shows that the linear relationship holds for a wide range of movements and force errors. Finally, because the forces are perpendicular to the movement, we will disregard the error parallel to the direction of movement, reducing Eq. 6 to a scalar equation. We are now in a position to write our system of equations in its final form:

$$\begin{cases} y^{(n)} = D_{k^{(n)}}(F^{(n)} - \hat{F}^{(n)}_{k^{(n)}}) \\ \hat{F}^{(m+1)}_l = \hat{F}^{(n)}_l + B_{l,k^{(n)}}\tilde{F}^{(n)} \quad l = 1, \ldots, 8 \end{cases} \tag{7}$$

Note that this is a system of nine equations: a single movement causes a change in all 8 directions for which the IM has an expectation. Let us now introduce a new variable $z^{(n)}_{k^{(n)}} \equiv D_{k^{(n)}}\hat{F}^{(n)}_{k^{(n)}}$, which represents the error (perpendicular displacement) that would have been experienced during this movement if we had not compensated for the expected field. With this substitution, Eq. 7 reduces to Eq. 1.

## 2.3 The shape of the generalization function $B$

Our task now is to give subjects a sequence of targets, observe the errors in their movements, and ask whether there are parameters for which the system of Eq. 7 gives a good fit. Given a sequence of $N$ movement directions, forces imposed on each movement, and the resulting errors ($\{k, F, y\}^{(n)}$, $j=1, \ldots, N$), we search for values of $B_{l,k}$, $D_k$ and initial conditions ($\hat{F}^{(0)}_m$, $m=1, \ldots, 8$) that minimize the squared difference, summed over the movements, between the $y$ calculated in Eq. 7 and the measured errors. One concern is that, in fitting a model with 80 parameters (64 from the $B$ matrix, 8 from $D$, and 8 from $\hat{F}^{(0)}$), we are likely to be overfitting our data. We address this concern by making the assumption that the $B$ matrix has a special shape: $B_{l,k} = b(\angle \dot{\mathbf{x}}_l \dot{\mathbf{x}}_k)$. That is, each entry in the $B$ matrix is determined according to the difference in angle between the two directions represented. This assumption implies that $\mathbf{g}(\dot{\mathbf{x}}_k)^T \mathbf{g}(\dot{\mathbf{x}}_l)$ depends only on $\angle \dot{\mathbf{x}}_k \dot{\mathbf{x}}_l$. This reduces the $B$ matrix to 8 parameters, and reduces the number of parameters in the model to 24.

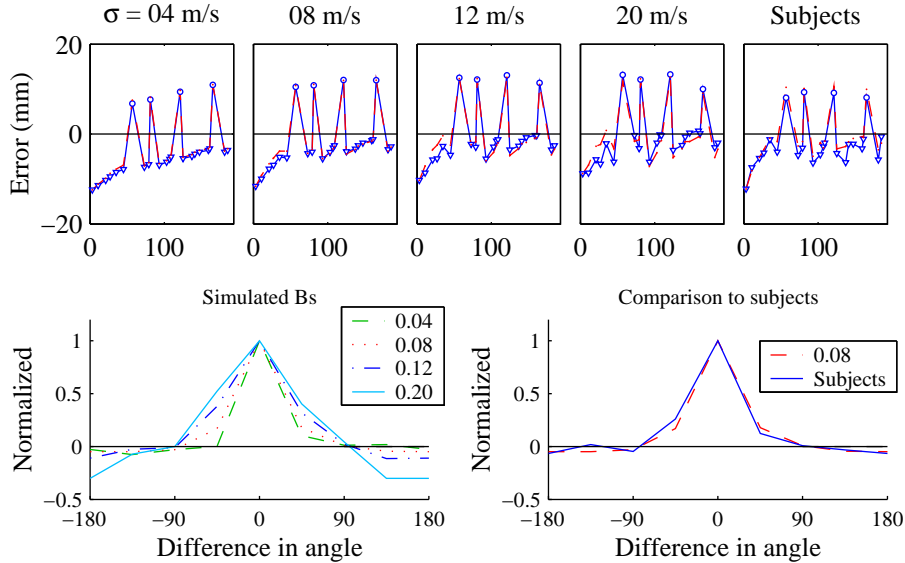

Figure 2: We simulated a system of equations representing dynamics of robot, human arm, and adaptive controller for movements to a total of 192 targets spanning 8 directions of movement. The adaptive controller learned by applying gradient descent ($\eta = 0.002$) to learn a gaussian basis encoding arm velocity with a $\sigma$ of 0.04, 0.08, 0.12, or 0.20 m/s. Errors, computed as displacement perpendicular to direction of target were measured at 250 msec and are plotted for one direction of movement (45 deg) (**a** - **d**). Simulated data is the solid line and the fit is shown as a dashed line. Circles indicate error on no field trials and triangles indicate error on fielded trials. The data for all 192 targets were then fit to Eq. 7 and the generalization matrix $B$ was estimated (**f**). Data was also collected from 76 subjects, and fit with the model (**e**), and it gave a generalization function that is nearly identicals to the generalization function of a controller using gaussians with a width of 0.08 m/s (**g**).

## 3   Results

We first tested the validity of our approach in an artificial learning system that used a simulation of human arm and robot dynamics to learn an IM of the imposed force field with gaussian basis elements. The result was a sequence of errors to a series of targets. We fit Eq. 7 to the sequence of errors and found an estimate for the generalization function (Fig. 2). As expected, when narrow basis elements are used, the generalization function is narrow. We next fit the same model to data that had been collected from 76 subjects and again found an excellent fit.

Plots **f** and **g** in Fig. 2 show the generalization function, $B$, as a function of the angle between $\dot{\mathbf{x}}_k$ and $\dot{\mathbf{x}}_l$. The demonstrate that errors in one direction affect movements in other directions both in simulations errors and in the subjects' errors. The greatest effect of error is in the direction in which the movement was made. The immediately neighboring directions are also significantly affected but the effect drops off with increasing distance. The generalization function which matched the human data was nearly identical to the one matching data produced by the simulation whose gaussians had $\sigma = 0.08$ m/sec.

The most interesting aspect of the success we had using the simple system in equation 7 to explain human behavior is that the global learning process is being charac-

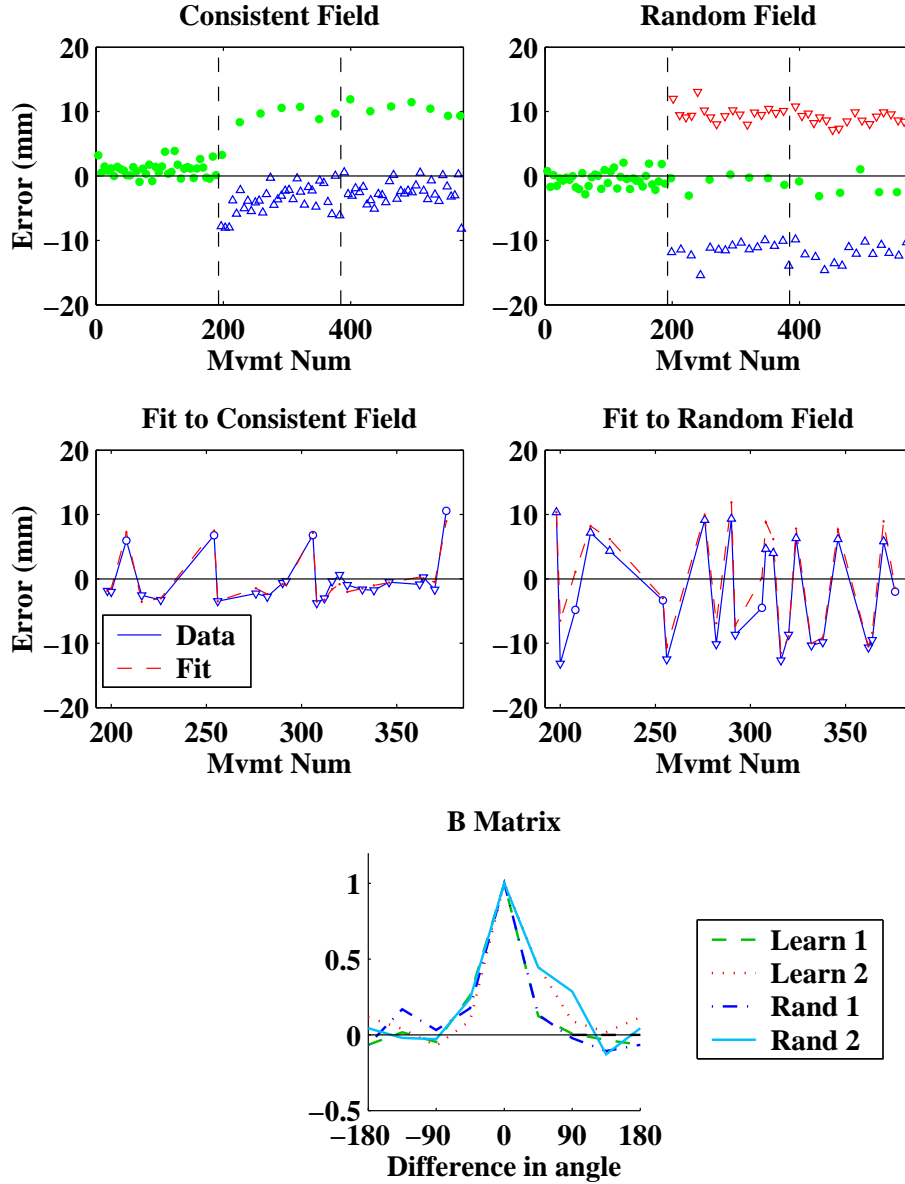

Figure 3: Fitting the model in Eq. 7 to a learning situation (**a** and **c**, 76 subjects) or a situation where subjects are presented with a random sequence of fields (**b** and **d**, 6 subjects) produce nearly identical models. **a** and **b** show errors (binned to 5 movements per data point), measured as perpendicular distance from a straight line trajectory at 250ms into the movement. Triangles are field A ($\mathbf{F} = [0\,13; -13\,0] \cdot \dot{\mathbf{x}}$) movements , wedges are field B ($\mathbf{F} = [0 - 13; 13\,0] \cdot \dot{\mathbf{x}}$), and filled circles are no field. The data is split into three sets of 192 movements. It can be seen that subjects in the learning paradigm learn to counteract the field, and show after affects. Subjects in the random field do not improve on either field, and do not show after affects. **c** and **d** show that the model fit both the learning paradigm and the random field paradigm. The fit is plotted for movements made to 90° during the first 192 movements following first exposure to the field (movements 193 through 384 in **a** and **b**). $r^2$ for the fits is 0.96 and 0.97 respectively. Fits to the last 192 movements in each paradigm gave $r^2$ of 0.96 and 0.98. Finally, in the bottom plot, we compare the generalization function, $B$, given by each fit. The normalized generalization function is nearly identical for the all four sets. The size of the central peak is 0.21 for both sets of the consistent field and 0.19 and 0.14, respectively, for the two sets of the random field.

terized as the accretion of small changes in the state of the controller accumulated over a large number of movements. In order to challenge this surprising aspect of the model, we decided to apply it to data in which human subjects performed movements in fields that varied randomly from trial to trial. In this case, no cumulative learning is possible. The important question is whether the model will still be able to fit the data. If it does fit the data, then the question is whether the parameters of the fit are similar to those derived from the learning paradigm.

Fig. 3 is a comparison of fitting a model to a consistent field and a random field. As seen in **a** and **b** of the figure, subjects are able to improve their performance through learning in a consistent field but they do not improve in the random field. However, as shown in in **c** and **d**, the model is able to fit the performance in both fields. Although the fits of each type of field were performed independently, we can see in **e** that the $B$ matrixes are nearly identical which indicates that trial-by-trial learning was the same for both types of fields. In the second set of the random paradigm, it seems as though the adjustment of state may slower. This raises the possibility that the process of movement-by-movement adjustment of state is gradually abandoned when it consistently fails to produce improvement. It is likely that in this case subjects come to rely on a feedback driven controller which would be unable to compensate for the errors generated early in the movement but would allow them to more quickly adjust to those errors as information about the field they are moving through is processed.

## 4    Conclusions

We hypothesized that the process of learning an internal model of the arm's dynamics may be similar to mechanisms of gradient descent in the framework of approximation theory. If so, then errors experienced in a given movement should affect subsequent movements in a meaningful way, and perhaps as simply as those predicted by the dynamical system in Eq. 7. These equations appear to fit both simulations and actual human data exceedingly well, making strong predictions about the shape of the basis with which the IM is apparently learned. Here we find that the shape of the basis remains invariant despite radical changes in pattern of errors, as exhibited when subjects were exposed to a random field as compared to a stationary field. We conclude that even when the task is unlearnable and errors approximate a flat line, the brain is attempting to learn with the same characteristic basis which is used when the task is simple and errors exponentially approach zero.

### References

[1] R. Shadmehr and F. A. Mussa-Ivaldi. Adaptive representation of dynamics during learning of a motor task. *J. Neurosci.*, 14(5 Pt 2):3208–3224, 1994.

[2] K. Thoroughman and R. Shadmehr. Learning of action through adaptive combination of motor primitives. *Nature*, 407(6805):742–747, 2000.

[3] R. A. Scheidt, J. B. Dingwell, and F. A. Mussa-Ivaldi. Learning to move amid uncertainty. *The Journal of Neurophysiology*, 86(2):971–985, 2001.

[4] R. M. Sanner and M. Kosha. A mathematical model of the adaptive control of human arm motions. *Biol. Cybern.*, 80(5):369–382, 1999.

[5] C. G. Atkeson. Learning arm kinematics and dynamics. *Annu. Rev. Neurosci.*, 12:157–183, 1989.

[6] Y. Uno, M. Kawato, and R. Suzuki. Formation and control of optimal trajectory in human multijoint arm movement. minimum torque-change model. *Biol. Cybern.*, 61(2):89–101, 1989.

[7] R. Shadmehr and H. H. Holcomb. Neural correlates of motor memory consolidation. *Science*, 277(5327):821–825, 1997.
